# Sparse Multinomial Logistic Regression via Bayesian L1 Regularisation

**Gavin C. Cawley**
School of Computing Sciences
University of East Anglia
Norwich, Norfolk, NR4 7TJ, U.K.
gcc@cmp.uea.ac.uk

**Nicola L. C. Talbot**
School of Computing Sciences
University of East Anglia
Norwich, Norfolk, NR4 7TJ, U.K.
nlct@cmp.uea.ac.uk

**Mark Girolami**
Department of Computing Science
University of Glasgow
Glasgow, Scotland, G12 8QQ, U.K.
girolami@dcs.gla.ac.uk

## Abstract

Multinomial logistic regression provides the standard penalised maximum-likelihood solution to multi-class pattern recognition problems. More recently, the development of sparse multinomial logistic regression models has found application in text processing and microarray classification, where explicit identification of the most informative features is of value. In this paper, we propose a sparse multinomial logistic regression method, in which the sparsity arises from the use of a Laplace prior, but where the usual regularisation parameter is integrated out analytically. Evaluation over a range of benchmark datasets reveals this approach results in similar generalisation performance to that obtained using cross-validation, but at greatly reduced computational expense.

## 1 Introduction

Multinomial logistic and probit regression are perhaps the classic statistical methods for multi-class pattern recognition problems (for a detailed introduction, see e.g. [1, 2]). The output of a multinomial logistic regression model can be interpreted as an *a-posteriori* estimate of the probability that a pattern belongs to each of $c$ disjoint classes. The probabilistic nature of the multinomial logistic regression model affords many practical advantages, such as the ability to set rejection thresholds [3], to accommodate unequal relative class frequencies in the training set and in operation [4], or to apply an appropriate loss matrix in making predictions that minimise the expected risk [5]. As a result, these models have been adopted in a diverse range of applications, including cancer classification [6, 7], text categorisation [8], analysis of DNA binding sites [9] and call routing. More recently, the focus of research has been on methods for inducing sparsity in (multinomial) logistic or probit regression models. In some applications, the identification of salient input features is of itself a valuable activity; for instance in cancer classification from micro-array gene expression data, the identification of *biomarker* genes, the pattern of expression of which is diagnostic of a particular form of cancer, may provide insight into the ætiology of the condition. In other applications, these methods are used to select a small number of basis functions to form a compact non-parametric classifier, from a set that may contain many thousands of candidate functions. In this case the sparsity is desirable for the purposes of computational expediency, rather than as an aid to understanding the data.

A variety of methods have been explored that aim to introduce sparsity in non-parametric regression models through the incorporation of a penalty or regularisation term within the training criterion. In the context of least-squares regression using Radial Basis Function (RBF) networks, Orr [10], proposes the use of local regularisation, in which a weight-decay regularisation term is used with distinct regularisation parameters for each weight. The optimisation of the Generalised Cross-Validation (GCV) score typically leads to the regularisation parameters for redundant basis functions achieving very high values, allowing them to be identified and pruned from the network (c.f. [11, 12]). The computational efficiency of this approach can be further improved via the use of Recursive Orthogonal Least Squares (ROLS). The relevance vector machine (RVM) [13] implements a form of Bayesian automatic relevance determination (ARD), using a separable Gaussian prior. In this case, the regularisation parameter for each weight is adjusted so as to maximise the marginal likelihood, also known as the Bayesian *evidence* for the model. An efficient component-wise training algorithm is given in [14]. An alternative approach, known as the LASSO [15], seeks to minimise the negative log-likelihood of the sample, subject to an upper bound on the sum of the absolute value of the weights (see also [16] for a practical training procedure). This strategy is equivalent to the use of a Laplace prior over the model parameters [17], which has been demonstrated to control over-fitting and induce sparsity in the weights of multi-layer perceptron networks [18]. The equivalence of the Laplace prior and a separable Gaussian prior (with appropriate choice of regularisation parameters) has been established by Grandvalet [11, 12], unifying these strands of research.

In this paper, we demonstrate that, in the case of the Laplace prior, the regularisation parameters can be integrated out analytically, obviating the need for a lengthy cross-validation based model selection stage. The resulting sparse multinomial logistic regression algorithm with Bayesian regularisation (SBMLR) is then fully automated and, having storage requirements that scale only linearly with the number of model parameters, is well suited to relatively large-scale applications. The remainder of this paper is set out as follows: The sparse multinomial logistic regression procedure with Bayesian regularisation is presented in Section 2. The proposed algorithm is then evaluated against competing approaches over a range of benchmark learning problems in Section 3. Finally, the work is summarised in Section 5 and conclusion drawn.

## 2  Method

Let $\mathcal{D} = \{(\boldsymbol{x}^n, \boldsymbol{t}^n)\}_{n=1}^{\ell}$ represent the training sample, where $\boldsymbol{x}^n \in \mathcal{X} \subset \mathbb{R}^d$ is the vector of input features for the $i^{\text{th}}$ example, and $\boldsymbol{t}^n \in \mathcal{T} = \{\boldsymbol{t} \mid \boldsymbol{t} \in \{0, 1\}^c, \|\boldsymbol{t}\|_1 = 1\}$ is the corresponding vector of desired outputs, using the usual 1-of-$c$ coding scheme. Multinomial logistic regression constructs a generalised linear model [1] with a *softmax* inverse link function [19], allowing the outputs to be interpreted as *a-posteriori* estimates of the probabilities of class membership,

$$p(t_i^n | \boldsymbol{x}^n) = y_i^n = \frac{\exp\{a_i^n\}}{\sum_{j=1}^c \exp\{a_j^n\}} \qquad \text{where} \qquad a_i^n = \sum_{j=1}^d w_{ij} x_j^n \tag{1}$$

Assuming that $\mathcal{D}$ represents an i.i.d. sample from a conditional multinomial distribution, then the negative log-likelihood, used as a measure of the data-misfit, can be written as,

$$E_{\mathcal{D}} = \sum_{n=1}^{\ell} E_{\mathcal{D}}^n = -\sum_{n=1}^{\ell} \sum_{i=1}^{c} t_i^n \log\{y_i^n\}$$

The parameters, $\boldsymbol{w}$ of the multinomial logistic regression model are given by the minimiser of a penalised maximum-likelihood training criterion,

$$L = E_{\mathcal{D}} + \alpha E_{\mathcal{W}} \qquad \text{where} \qquad E_{\mathcal{W}} = \sum_{i=1}^{c} \sum_{j=1}^{d} |w_{ij}| \tag{2}$$

and $\alpha$ is a regularisation parameter [20] controlling the bias-variance trade-off [21]. At a minima of $L$, the partial derivatives of $L$ with respect to the model parameters will be uniformly zero, giving

$$\left| \frac{\partial E_{\mathcal{D}}}{\partial w_{ij}} \right| = \alpha \quad \text{if } |w_{ij}| > 0 \qquad \text{and} \qquad \left| \frac{\partial E_{\mathcal{D}}}{\partial w_{ij}} \right| < \alpha \quad \text{if } |w_{ij}| = 0.$$

This implies that if the sensitivity of the negative log-likelihood with respect to a model parameter, $w_{ij}$, falls below $\alpha$, then the value of that parameter will be set exactly to zero and the corresponding input feature can be pruned from the model.

## 2.1 Eliminating the Regularisation Parameters

Minimisation of (2) has a straight-forward Bayesian interpretation; the posterior distribution for $\boldsymbol{w}$, the parameters of the model given by (1), can be written as

$$p(\boldsymbol{w}|\mathcal{D}) \propto P(\mathcal{D}|\boldsymbol{w})P(\boldsymbol{w}).$$

$L$ is then, up to an additive constant, the negative logarithm of the posterior density. The prior over model parameters, $\boldsymbol{w}$, is then given by a separable Laplace distribution

$$P(\boldsymbol{w}) = \left(\frac{\alpha}{2}\right)^W \exp\{-\alpha E_{\mathcal{W}}\} = \prod_{i=1}^{W} \frac{\alpha}{2} \exp\{-\alpha|w_i|\}, \tag{3}$$

where $W$ is the number of active (non-zero) model parameters. A good value for the regularisation parameter $\alpha$ can be estimated, within a Bayesian framework, by maximising the *evidence* [22] or alternatively it may be integrated out analytically [17, 23]. Here we take the latter approach, where the prior distribution over model parameters is given by marginalising over $\alpha$,

$$p(\boldsymbol{w}) = \int p(\boldsymbol{w}|\alpha)p(\alpha)d\alpha.$$

As $\alpha$ is a scale parameter, an appropriate ignorance prior is given by the improper Jeffrey's prior, $p(\alpha) \propto 1/\alpha$, corresponding to a uniform prior over $\log \alpha$. Substituting equation (3) and noting that $\alpha$ is strictly positive,

$$p(\boldsymbol{w}) = \frac{1}{2^W} \int_0^\infty \alpha^{W-1} \exp\{-\alpha E_{\mathcal{W}}\}d\alpha.$$

Using the Gamma integral, $\int_0^\infty x^{\nu-1}e^{-\mu x}dx = \frac{\Gamma(\nu)}{\mu^\nu}$ [24, equation 3.384], we obtain

$$p(\boldsymbol{w}) = \frac{1}{2^W} \frac{\Gamma(W)}{E_{\mathcal{W}}^W} \qquad \Longrightarrow \qquad -\log p(\boldsymbol{w}) \propto W \log E_{\mathcal{W}},$$

giving a revised optimisation criterion for sparse logistic regression with Bayesian regularisation,

$$M = E_{\mathcal{D}} + W \log E_{\mathcal{W}}, \tag{4}$$

in which the regularisation parameter has been eliminated, for further details and theoretical justification, see [17]. Note that we integrate out the regularisation parameter and optimise the model parameters, which is unusual in that most Bayesian approaches, such as the relevance vector machine [13] optimise the regularisation parameters and integrate over the weights.

### 2.1.1 Practical Implementation

The training criterion incorporating a fully Bayesian regularisation term can be minimised via a simple modification of existing cyclic co-ordinate descent algorithms for sparse regression using a Laplace prior (e.g. [25, 26]). Differentiating the original and modified training criteria, (2) and (4) respectively, we have that

$$\nabla L = \nabla E_{\mathcal{D}} + \alpha \nabla E_{\mathcal{W}} \qquad \text{and} \qquad \nabla M = \nabla E_{\mathcal{D}} + \tilde{\alpha} \nabla E_{\mathcal{W}}$$

where

$$1/\tilde{\alpha} = \frac{1}{W} \sum_{i=1}^{W} |w_i|. \tag{5}$$

From a gradient descent perspective, minimising $M$ effectively becomes equivalent to minimising $L$, assuming that the regularisation parameter, $\alpha$, is continuously updated according to (5) following every change in the vector of model parameters, $\boldsymbol{w}$ [17]. This requires only a very minor modification of the existing training algorithm, whilst eliminating the only training parameter and hence the need for a model selection procedure in fitting the model.

### 2.1.2 Equivalence of Marginalisation and Optimisation under the Evidence Framework

Williams [17] notes that, at least in the case of the Laplace prior, integrating out the regularisation parameter analytically is equivalent to its optimisation under the evidence framework of MacKay [22]. The argument provided by Williams can be summarised as follows: The evidence framework sets the value of the regularisation parameter so as to optimise the marginal likelihood,

$$P(\mathcal{D}) = \int P(\mathcal{D}|\boldsymbol{w})P(\boldsymbol{w})d\boldsymbol{w},$$

also known as the *evidence* for the model. The Bayesian interpretation of the regularised objective function gives,

$$P(\mathcal{D}) = \frac{1}{Z_{\mathcal{W}}} \int \exp\{-L\} \, d\boldsymbol{w},$$

where $Z_{\mathcal{W}}$ is a normalising constant for the prior over the model parameters, for the Laplace prior, $Z_{\mathcal{W}} = (2/\alpha)^W$. In the case of multinomial logistic regression, $E_{\mathcal{D}}$ represents the negative logarithm of a normalised distribution, and so the corresponding normalising constant for the data misfit term is redundant. Unfortunately this integral is analytically intractable, and so we adopt the Laplace approximation, corresponding to a Gaussian posterior distribution for the model parameters, centred on their most probable value, $\boldsymbol{w}^{\mathrm{MP}}$,

$$L(\boldsymbol{w}) = L(\boldsymbol{w}^{\mathrm{MP}}) + \frac{1}{2}(\boldsymbol{w} - \boldsymbol{w}^{\mathrm{MP}})^T \boldsymbol{A} (\boldsymbol{w} - \boldsymbol{w}^{\mathrm{MP}})$$

where $\boldsymbol{A} = \nabla\nabla L$ is the Hessian of the regularised objective function. The regulariser corresponding to the Laplace prior is locally a hyper-plane, and so does not contribute to the Hessian and so $\boldsymbol{A} = \nabla\nabla E_{\mathcal{D}}$. The negative logarithm of the evidence can then be written as,

$$-\log P(\mathcal{D}) = E_{\mathcal{D}} + \alpha E_{\mathcal{W}} + \frac{1}{2}\log|\boldsymbol{A}| + \log Z_{\mathcal{W}} + \text{constant}.$$

Setting the derivative of the evidence with respect to $\alpha$ to zero, gives rise to a simple update rule for the regularisation parameter,

$$\frac{1}{\tilde{\alpha}} = \frac{1}{W}\sum_{j=1}^{W}|w_j|,$$

which is equivalent to the update rule obtained using the integrate-out approach. Maximising the evidence for the model also provides a convenient means for model selection. Using the Laplace approximation, evidence for a multinomial logistic regression model under the proposed Bayesian regularisation scheme is given by

$$-\log\{\mathcal{D}\} = E_{\mathcal{D}} + W\log E_{\mathcal{W}} - \log\left\{\frac{\Gamma(W)}{2^W}\right\} + \frac{1}{2}\log|\boldsymbol{A}| + \text{constant}$$

where $\boldsymbol{A} = \nabla\nabla L$.

### 2.2 A Simple but Efficient Training Algorithm

In this study, we adopt a simplified version of the efficient component-wise training algorithm of Shevade and Keerthi [25], adapted for multinomial, rather than binomial, logistic regression. The principal advantage of a component-wise optimisation algorithm is that the Hessian matrix is not required, but only the first and second partial derivatives of the regularised training criterion. The first partial derivatives of the data mis-fit term are given by,

$$\frac{\partial E_{\mathcal{D}}^n}{\partial a_j^n} = \sum_{i=1}^{c} \frac{\partial E_{\mathcal{D}}^n}{\partial y_i^n}\frac{\partial y_i^n}{\partial a_j^n} \qquad \text{where} \qquad \frac{\partial E_{\mathcal{D}}^n}{\partial y_i^n} = -\frac{t_i^n}{y_i^n}, \qquad \frac{\partial y_i^n}{\partial a_j^n} = y_i\delta_{ij} - y_iy_j$$

and $\delta_{ij} = 1$ if $i = j$ and otherwise $\delta_{ij} = 0$. Substituting, we obtain,

$$\frac{\partial E_{\mathcal{D}}}{\partial \boldsymbol{a}_i} = \sum_{n=1}^{\ell}[y_i^n - t_i^n] \quad \Longrightarrow \quad \frac{\partial E_{\mathcal{D}}}{\partial w_{ij}} = \sum_{n=1}^{\ell}[y_i^n - t_i^n]\,x_j^n = \sum_{n=1}^{\ell}y_i^n x_j^n - \sum_{n=1}^{\ell}t_i^n x_j^n.$$

Similarly, the second partial derivatives are given by,

$$\frac{\partial^2 E_{\mathcal{D}}}{\partial w_{ij}} = \sum_{n=1}^{\ell} x_j^n \frac{\partial y_i^n}{\partial w_{ij}} = \sum_{n=1}^{\ell} y_i^n \left(1 - y_i^n\right) \left[x_j^n\right]^2.$$

The Laplace regulariser is locally a hyperplane, with the magnitude of the gradient given by the regularisation parameter, $\alpha$,

$$\frac{\partial \alpha E_{\mathcal{W}}}{\partial w_{ij}} = \text{sign}\left\{w_{ij}\right\}\alpha \qquad \text{and} \qquad \frac{\partial^2 \alpha E_{\mathcal{W}}}{\partial w_{ij}^2} = 0.$$

The partial derivatives of the regularisation term are not defined at the origin, and so we define the *effective* gradient of the regularised loss function as follows:

$$\frac{\partial L}{\partial w_{ij}} = \begin{cases} \frac{\partial E_{\mathcal{D}}}{\partial w_{ij}} + \alpha & \text{if } w_{ij} > 0 \\ \frac{\partial E_{\mathcal{D}}}{\partial w_{ij}} - \alpha & \text{if } w_{ij} < 0 \\ \frac{\partial E_{\mathcal{D}}}{\partial w_{ij}} + \alpha & \text{if } w_{ij} = 0 \text{ and } \frac{\partial E_{\mathcal{D}}}{\partial w_{ij}} + \alpha < 0 \\ \frac{\partial E_{\mathcal{D}}}{\partial w_{ij}} - \alpha & \text{if } w_{ij} = 0 \text{ and } \frac{\partial E_{\mathcal{D}}}{\partial w_{ij}} - \alpha > 0 \\ 0 & \text{otherwise} \end{cases}$$

Note that the value of a weight may be stable at zero if the derivative of the regularisation term dominates the derivative of the data misfit. The parameters of the model may then be optimised, using Newton's method, i.e.

$$w_{ij} \leftarrow w_{ij} - \frac{\partial E_{\mathcal{D}}}{\partial w_{ij}} \left[\frac{\partial^2 E_{\mathcal{D}}}{\partial w_{ij}^2}\right]^{-1}.$$

Any step that causes a change of sign in a model parameter is truncated and that parameter set to zero. All that remains is to decide on a heuristic used to select the parameter to be optimised in each step. In this study, we adopt the heuristic chosen by Shevade and Keerthi, in which the parameter having the steepest gradient is selected in each iteration. The optimisation proceeds using two nested loops, in the inner loop, only active parameters are considered. If no further progress can be made by optimising active parameters, the search is extended to parameters that are currently set to zero. An optimisation strategy based on scaled conjugate gradient descent [27] has also be found to be effective.

## 3 Results

The proposed sparse multinomial logistic regression method incorporating Bayesian regularisation using a Laplace prior (SBMLR) was evaluated over a suite of well-known benchmark datasets, against sparse multinomial logistic regression with five-fold cross-validation based optimisation of the regularisation parameter using a simple line search (SMLR). Table 1 shows the test error rate and cross-entropy statistics for SMLR and SBMLR methods over these datasets. Clearly, there is little reason to prefer either model over the other in terms of generalisation performance, as neither consistently dominates the other, either in terms of error rate or cross-entropy. Table 1 also shows that the Bayesian regularisation scheme results in models with a slightly higher degree of sparsity (i.e. the proportion of weights pruned from the model). However, the most striking aspect of the comparison is that the Bayesian regularisation scheme is typically around two orders of magnitude faster than the cross-validation based approach, with SBMLR being approximately five times faster in the worst case (COVTYPE).

### 3.1 The Value of Probabilistic Classification

Probabilistic classifiers, i.e. those that providing an *a-posteriori* estimate of the probability of class membership, can be used in minimum risk classification, using an appropriate loss matrix to account for the relative costs of different types of error. Probabilistic classifiers allow rejection thresholds to be set in a straight-forward manner. This is particularly useful in a medical setting, where it may be prudent to refer a patient for further tests if the diagnosis is uncertain. Finally, the output of

Table 1: Evaluation of linear sparse multinomial logistic regression methods over a set of nine benchmark datasets. The best results for each statistic are shown in bold. The final column shows the logarithm of the ratio of the training times for the SMLR and SBMLR, such that a value of 2 would indicate that SBMLR is 100 times faster than SMLR for a given benchmark dataset.

| Benchmark | Error Rate | | Cross Entropy | | Sparsity | | $\log_{10} \frac{T_{\text{SMLR}}}{T_{\text{SBMLR}}}$ |
|---|---|---|---|---|---|---|---|
| | SBMLR | SMLR | SBMLR | SMLR | SBMLR | SMLR | |
| **Covtype** | 0.4051 | **0.4041** | **0.9590** | 0.9733 | **0.4312** | 0.3069 | 0.6965 |
| **Crabs** | **0.0350** | 0.0500 | 0.1075 | **0.0891** | **0.2708** | 0.0635 | 2.7949 |
| **Glass** | 0.3318 | **0.3224** | **0.9398** | 0.9912 | 0.4400 | **0.4700** | 1.9445 |
| **Iris** | **0.0267** | **0.0267** | **0.0792** | 0.0867 | **0.4067** | **0.4067** | 1.9802 |
| **Isolet** | **0.0475** | 0.0513 | **0.1858** | 0.2641 | **0.9311** | 0.8598 | 1.3110 |
| **Satimage** | 0.1610 | **0.1600** | 0.3717 | **0.3708** | **0.3694** | 0.2747 | 1.3083 |
| **Viruses** | **0.0328** | **0.0328** | 0.1670 | **0.1168** | **0.8118** | 0.7632 | 2.1118 |
| **Waveform** | **0.1290** | 0.1302 | **0.3124** | 0.3131 | 0.3712 | **0.3939** | 1.8133 |
| **Wine** | **0.0225** | 0.0281 | 0.0827 | **0.0825** | **0.6071** | 0.5524 | 2.5541 |

a probabilistic classifier can be adjusted *after* training to compensate for a difference between the relative class frequencies in the training set and those observed in operation. Saerens [4] provides a simple expectation-maximisation (EM) based procedure for estimating unknown operational *a-priori* probabilities from the output of a probabilistic classifier (c.f. [28]). Let $p_t(\mathcal{C}_i)$ represent the *a-priori* probability of class $\mathcal{C}_i$ in the training set and $p_t(\mathcal{C}_i|\boldsymbol{x}^n)$ represent the raw output of the classifier for the $n^{\text{th}}$ pattern of the test data (representing operational conditions). The operational *a-priori* probabilities, $p_o(\mathcal{C}_i)$ can then be updated iteratively via

$$p_o^{(s)}(\omega_i|\boldsymbol{x}^n) = \frac{\frac{p_o^{(s)}(\omega_i)}{p_t(\omega_i)}p_t(\omega_i|\boldsymbol{x}^n)}{\sum_{j=1}^{c}\frac{p_o^{(s)}(\omega_j)}{p_t(\omega_j)}p_t(\omega_j|\boldsymbol{x}^n)} \qquad \text{and} \qquad p_o^{(s+1)}(\omega_i) = \frac{1}{\ell}\sum_{n=1}^{N}p_o^{(s)}(\omega_i|\boldsymbol{x}^n), \qquad (6)$$

beginning with $p_o^{(0)}(\mathcal{C}_i) = p_t(\mathcal{C}_i)$. Note that the labels of the test examples are not required for this procedure. The adjusted estimates of *a-posteriori* probability are then given by the first part of equation (6). The training and validation sets of the COVTYPE benchmark have been artificially balanced, by random sampling, so that each class is represented by the same number of examples. The test set consists of the unused patterns, and so the test set *a-priori* probabilities are both highly disparate and very different from the training set *a-priori* probabilities. Figure 1 and Table 2 summarise the results obtained using the raw and corrected outputs of a linear SBMLR model on this dataset, clearly demonstrating a key advantage of probabilistic classifiers over purely discriminative methods, for example the support vector machine (note the same procedure could be applied to the SMLR model with similar results).

Table 2: Error rate and average cross-entropy score for linear SBMLR models of the COVTYPE benchmark, using the raw and corrected outputs.

| Statistic | Raw | Corrected |
|---|---|---|
| **Error Rate** | 40.51% | 28.57% |
| **Cross-Entropy** | 0.9590 | 0.6567 |

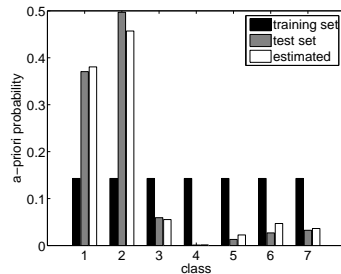

Figure 1: Training set, test set and estimated *a-priori* probabilities for the COVTYPE benchmark.

## 4  Relationship to Existing Work

The sparsity inducing Laplace density has been utilized previously in [15, 25, 26, 29–31] and emerges as the marginal of a scale-mixture-of-Gaussians where the corresponding prior is an Exponential such that

$$\int \mathcal{N}_w(0, \tau)\mathcal{E}_\tau(\gamma)d\tau = \frac{\alpha}{2}\exp(-\alpha|w|)$$

where $\mathcal{E}_\tau(\gamma)$ is an Exponential distribution over $\tau$ with parameter $\gamma$ and $\alpha = \sqrt{\gamma}$. In [29] this hierarchical representation of the Laplace prior is utilized to develop an EM style sparse binomial probit regression algorithm. The hyper-parameter $\alpha$ is selected via cross-validation but in an attempt to circumvent this requirement a Jeffreys prior is placed on $\tau$ and is used to replace the exponential distribution in the above integral. This yields an improper parameter free prior distribution over $w$ which removes the explicit requirement to perform any cross-validation. However, the method developed in [29] is restricted to binary classification and has compute scaling $\mathcal{O}(d^3)$ which prohibits its use on moderately high-dimensional problems.

Likewise in [13] the RVM employs a similar scale-mixture for the prior where now the Exponential distribution is replaced by a Gamma distribution whose marginal yields a Student prior distribution. No attempt is made to estimate the associated hyper-parameters and these are typically set to zero producing, as in [29], a sparsity inducing improper prior. As with [29] the original scaling of [13] is, at worst, $\mathcal{O}(d^3)$, though more efficient methods have been developed in [14]. However the analysis holds only for a binary classifier and it would be non-trivial to extend this to the multi-class domain.

A similar multinomial logistic regression model to the one proposed in this paper is employed in [26] where the algorithm is applied to large scale classification problems and yet they, as with [25], have to resort to cross-validation in obtaining a value for the hyper-parameters of the Laplace prior.

## 5  Summary

In this paper we have demonstrated that the regularisation parameter used in sparse multinomial logistic regression using a Laplace prior can be integrated out analytically, giving similar performance in terms of generalisation as is obtained using extensive cross-validation based model selection, but at a greatly reduced computational expense. It is interesting to note that the SBMLR implements a strategy that is exactly the opposite of the relevance vector machine (RVM) [13], in that it integrates over the hyper-parameter and optimises the weights, rather than marginalising the model parameters and optimising the hyper-parameters. It seems reasonable to suggest that this approach is feasible in the case of the Laplace prior as the pruning action of this prior ensures that values of all of the weights are strongly determined by the data misfit term. A similar strategy has already proved effective in cancer classification based on gene expression microarray data in a binomial setting [32], and we plan to extend this work to multi-class cancer classification in the near future.

## Acknowledgements

The authors thank the anonymous reviewers for their helpful and constructive comments. MG is supported by EPSRC grant EP/C010620/1.

## References

[1] P. McCullagh and J. A. Nelder. *Generalized linear models*, volume 37 of *Monographs on Statistics and Applied Probability*. Chapman & Hall/CRC, second edition, 1989.

[2] D. W. Hosmer and S. Lemeshow. *Applied logistic regression*. Wiley, second edition, 2000.

[3] C. K. Chow. On optimum recognition error and reject tradeoff. *IEEE Transactions on Information Theory*, 16(1):41–46, January 1970.

[4] M. Saerens, P. Latinne, and C. Decaestecker. Adjusting the outputs of a classifier to new a priori probabilities: A simple procedure. *Neural Computation*, 14(1):21–41, 2001.

[5] J. O. Berger. *Statistical decision theory and Bayesian analysis*. Springer Series in Statistics. Springer, second edition, 1985.

[6] J. Zhu and T. Hastie. Classification of gene microarrays by penalized logistic regression. *Biostatistics*, 5(3):427–443, 2004.

[7] X. Zhou, X. Wang, and E. R. Dougherty. Multi-class cancer classification using multinomial probit regression with Bayesian gene selection. *IEE Proceedings - Systems Biology*, 153(2):70–76, March 2006.

[8] T. Zhang and F. J. Oles. Text categorization based on regularised linear classification methods. *Information Retrieval*, 4(1):5–31, April 2001.

[9] L. Narlikar and A. J. Hartemink. Sequence features of DNA binding sites reveal structural class of associated transcription factor. *Bioinformatics*, 22(2):157–163, 2006.

[10] M. J. L. Orr. Regularisation in the selection of radial basis function centres. *Neural Computation*, 7(3):606–623, 1995.

[11] Y. Grandvalet. Least absolute shrinkage is equivalent to quadratic penalisation. In L. Niklasson, M. Bodén, and T. Ziemske, editors, *Proceedings of the International Conference on Artificial Neural Networks*, Perspectives in Neural Computing, pages 201–206, Skövde, Sweeden, September 2–4 1998. Springer.

[12] Y. Grandvalet and S. Canu. Outcomes of the quivalence of adaptive ridge with least absolute shrinkage. In *Advances in Neural Information Processing Systems*, volume 11. MIT Press, 1999.

[13] M. E. Tipping. Sparse Bayesian learning and the Relevance Vector Machine. *Journal of Machine Learning Research*, 1:211–244, 2001.

[14] A. C. Faul and M. E. Tipping. Fast marginal likelihood maximisation for sparse Bayesian models. In C. M. Bishop and B. J. Frey, editors, *Proceedings of the Ninth International Workshop on Artificial Intelligence and Statistics*, Key West, FL, USA, 3–6 January 2003.

[15] R. Tibshirani. Regression shrinkage and selection via the LASSO. *Journal of the Royal Statistical Society - Series B*, 58:267–288, 1996.

[16] B. Efron, T. Hastie, I. Johnstone, and R. Tibshirani. Least angle regression. *Annals of Statistics*, 32(2):407–499, 2004.

[17] P. M. Williams. Bayesian regularization and pruning using a Laplace prior. *Neural Computation*, 7(1):117–143, 1995.

[18] C. M. Bishop. *Neural networks for pattern recognition*. Oxford University Press, 1995.

[19] J. S. Bridle. Probabilistic interpretation of feedforward classification network outputs, with relationships to statistical pattern recognition. In F. Fogelman Soulié and J. Hérault, editors, *Neurocomputing: Algorithms, architectures and applications*, pages 227–236. Springer-Verlag, New York, 1990.

[20] A. N. Tikhonov and V. Y. Arsenin. *Solutions of ill-posed problems*. John Wiley, New York, 1977.

[21] S. Geman, E. Bienenstock, and R. Doursat. Neural networks and the bias/variance dilema. *Neural Computation*, 4(1):1–58, 1992.

[22] D. J. C. MacKay. The evidence framework applied to classification networks. *Neural Computation*, 4(5):720–736, 1992.

[23] W. L. Buntine and A. S. Weigend. Bayesian back-propagation. *Complex Systems*, 5:603–643, 1991.

[24] I. S. Gradshteyn and I. M. Ryzhic. *Table of Integrals, Series and Products*. Academic Press, fifth edition, 1994.

[25] S. K. Shevade and S. S. Keerthi. A simple and efficient algorithm for gene selection using sparse logistic regression. *Bioinformatics*, 19(17):2246–2253, 2003.

[26] D. Madigan, A. Genkin, D. D. Lewis, and D. Fradkin. Bayesian multinomial logistic regression for author identification. In *AIP Conference Proceedings*, volume 803, pages 509–516, 2005.

[27] P. M. Williams. A Marquardt algorithm for choosing the step size in backpropagation learning with conjugate gradients. Technical Report CSRP-229, University of Sussex, February 1991.

[28] G. J. McLachlan. *Discriminant analysis and statistical pattern recognition*. Wiley, 1992.

[29] M. Figueiredo. Adaptive sparseness for supervised learning. *IEEE Transactions on Pattern Analysis and Machine Intelligence*, 25(9):1150–1159, September 2003.

[30] B. Krishnapuram, L. Carin, M. A. T. Figueiredo, and A. J. Hartemink. Sprse multinomial logistic regression: Fast algorithms and generalisation bounds. *IEEE Transactions on Pattern Analysis and Machine Intelligence*, 27(6):957–968, June 2005.

[31] J. M. Bioucas-Dias, M. A. T. Figueiredo, and J. P. Oliveira. Adaptive total variation image deconvolution: A majorization-minimization approach. In *Proceedings of the European Signal Processing Conference (EUSIPCO'2006)*, Florence, Italy, September 2006.

[32] G. C. Cawley and N. L. C. Talbot. Gene selection in cancer classification using sparse logistic regression with Bayesian regularisation. *Bioinformatics*, 22(19):2348–2355, October 2006.
